# Linear Complementarity for Regularized Policy Evaluation and Improvement

**Jeff Johns**       **Christopher Painter-Wakefield**       **Ronald Parr**
Department of Computer Science
Duke University
Durham, NC 27708
{johns, paint007, parr}@cs.duke.edu

## Abstract

Recent work in reinforcement learning has emphasized the power of $L_1$ regularization to perform feature selection and prevent overfitting. We propose formulating the $L_1$ regularized linear fixed point problem as a linear complementarity problem (LCP). This formulation offers several advantages over the LARS-inspired formulation, LARS-TD. The LCP formulation allows the use of efficient off-the-shelf solvers, leads to a new uniqueness result, and can be initialized with starting points from similar problems (warm starts). We demonstrate that warm starts, as well as the efficiency of LCP solvers, can speed up policy iteration. Moreover, warm starts permit a form of modified policy iteration that can be used to approximate a "greedy" homotopy path, a generalization of the LARS-TD homotopy path that combines policy evaluation and optimization.

## 1   Introduction

$L_1$ regularization has become an important tool over the last decade with a wide variety of machine learning applications. In the context of linear regression, its use helps prevent overfitting and enforces sparsity in the problem's solution. Recent work has demonstrated how $L_1$ regularization can be applied to the value function approximation problem in Markov decision processes (MDPs). Kolter and Ng [1] included $L_1$ regularization within the least-squares temporal difference learning [2] algorithm as LARS-TD, while Petrik et al. [3] adapted an approximate linear programming algorithm. In both cases, $L_1$ regularization automates the important task of selecting relevant features, thereby easing the design choices made by a practitioner.

LARS-TD provides a homotopy method for finding the $L_1$ regularized linear fixed point formulated by Kolter and Ng. We reformulate the $L_1$ regularized linear fixed point as a linear complementarity problem (LCP). This formulation offers several advantages. It allows us to draw upon the rich theory of LCPs and optimized solvers to provide strong theoretical guarantees and fast performance. In addition, we can take advantage of the "warm start" capability of LCP solvers to produce algorithms that are better suited to the sequential nature of policy improvement than LARS-TD, which must start from scratch for each new policy.

## 2   Background

First, we introduce MDPs and linear value function approximation. We then review $L_1$ regularization and feature selection for regression problems. Finally, we introduce LCPs. We defer discussion of $L_1$ regularization and feature selection for reinforcement learning (RL) until section 3.

## 2.1 MDP and Value Function Approximation Framework

We aim to discover optimal, or near-optimal, policies for Markov decision processes (MDPs) defined by the quintuple $M = (S, A, P, R, \gamma)$. Given a state $s \in S$, the probability of a transition to a state $s' \in S$ when action $a \in A$ is taken is given by $P(s'|s, a)$. The reward function is a mapping from states to real numbers $R : S \mapsto \mathbb{R}$. A policy $\pi$ for $M$ is a mapping from states to actions $\pi : s \mapsto a$ and the transition matrix induced by $\pi$ is denoted $P^\pi$. Future rewards are discounted by $\gamma \in [0, 1)$.

The value function at state $s$ for policy $\pi$ is the expected total $\gamma$-discounted reward for following $\pi$ from $s$. In matrix-vector form, this is written:

$$V^\pi = T^\pi V^\pi = R + \gamma P^\pi V^\pi,$$

where $T^\pi$ is the *Bellman operator* for policy $\pi$ and $V^\pi$ is the fixed point of this operator. An optimal policy, $\pi^*$, maximizes state values, has value function $V^*$, and is the fixed point of the $T^*$ operator:

$$T^*V(s) = R(s) + \gamma \max_{a \in A} \sum_{s' \in S} P(s'|s, a)V(s').$$

Of the many algorithms that exist for finding $\pi^*$, *policy iteration* is most relevant to the presentation herein. For any policy $\pi_j$, policy iteration computes $V^{\pi_j}$, then determines $\pi_{j+1}$ as the "greedy" policy with respect to $V^{\pi_j}$:

$$\pi_{j+1}(s) = \arg\max_{a \in A}[R(s) + \gamma \sum_{s' \in S} P(s'|s, a)V^{\pi_j}(s')].$$

This is repeated until some convergence condition is met. For an exact representation of each $V^{\pi_j}$, the algorithm will converge to an optimal policy and the unique, optimal value function $V^*$.

The value function, transition model, and reward function are often too large to permit an exact representation. In such cases, an approximation architecture is used for the value function. A common choice is $\hat{V} = \Phi w$, where $w$ is a vector of $k$ scalar weights and $\Phi$ stores a set of $k$ features in an $n \times k$ matrix with one row per state. Since $n$ is often intractably large, $\Phi$ can be thought of as populated by $k$ linearly independent *basis functions*, $\varphi_1 \ldots \varphi_k$, implicitly defining the columns of $\Phi$.

For the purposes of estimating $w$, it is common to replace $\Phi$ with $\hat{\Phi}$, which samples rows of $\Phi$, though for conciseness of presentation we will use $\Phi$ for both, since algorithms for estimating $w$ are essentially identical if $\hat{\Phi}$ is substituted for $\Phi$. Typical linear function approximation algorithms [2] solve for the $w$ which is a fixed point:

$$\Phi w = \Pi(R + \gamma \Phi'^\pi w) = \Pi T^\pi \Phi w,$$

where $\Pi$ is the $L_2$ projection into the span of $\Phi$ and $\Phi'^\pi$ is $P^\pi \Phi$ in the explicit case and composed of sampled next features in the sampled case. Likewise, we overload $T^\pi$ for the sampled case.

## 2.2 $L_1$ Regularization and Feature Selection in Regression

In regression, the $L_1$ regularized least squares problem is defined as:

$$w = \arg\min_{x \in \mathbb{R}^k} \frac{1}{2}\|\Phi x - y\|_2^2 + \beta\|x\|_1, \tag{1}$$

where $y \in \mathbb{R}^n$ is the target function and $\beta \in \mathbb{R}_{\geq 0}$ is a regularization parameter. This penalized regression problem is equivalent to the *Lasso* [4], which minimizes the squared residual subject to a constraint on $\|x\|_1$. The use of the $L_1$ norm in the objective function prevents overfitting, but also serves a secondary purpose of promoting sparse solutions (i.e., coefficients $w$ containing many 0s). Therefore, we can think of $L_1$ regularization as performing *feature selection*. The Lasso's objective function is convex, ensuring the existence of a global (though not necessarily unique) minimum.

Even though the optimal solution to the Lasso can be computed in a fairly straightforward manner using convex programming, this approach is not very efficient for large problems. This is a motivating factor for the least angle regression (LARS) algorithm [5], which can be thought of as a homotopy method for solving the Lasso for *all* nonnegative values of $\beta$. We do not repeat the details of the algorithm here, but point out that this is easier than it might sound at first because the homotopy path in $\beta$-space is *piecewise linear* (with finitely many segments). Furthermore, there exists a closed form solution for moving from one piecewise linear segment to the next segment. An important benefit of LARS is that it provides solutions for all values of $\beta$ in a single run of the algorithm. Cross-validation can then be performed to select an appropriate value.

## 2.3  LCP and BLCP

Given a square matrix $M$ and a vector $q$, a linear complementarity problem (LCP) seeks vectors $w \geq \mathbf{0}$ and $z \geq \mathbf{0}$ with $w^T z = 0$ and

$$w = q + Mz.$$

The problem is thus parameterized by $\mathrm{LCP}(q, M)$. Even though LCPs may appear to be simple feasibility problems, the framework is rich enough to express any convex quadratic program.

The *bounded* linear complementarity problem (BLCP) [6] includes box constraints on $z$. The BLCP computes $w$ and $z$ where $w = q + Mz$ and each variable $z_i$ meets one of the following conditions:

$$
\begin{align}
z_i = u_i \quad &\Longrightarrow \quad w_i \leq 0 \tag{2a}\\
z_i = l_i \quad &\Longrightarrow \quad w_i \geq 0 \tag{2b}\\
l_i < z_i < u_i \quad &\Longrightarrow \quad w_i = 0 \tag{2c}
\end{align}
$$

with bounds $-\infty \leq l_i < u_i \leq \infty$. The parameterization is written $\mathrm{BLCP}(q, M, l, u)$. Notice that an LCP is a special case of a BLCP with $l_i = 0$ and $u_i = \infty, \forall i$. Like the LCP, the BLCP has a unique solution when $M$ is a P-matrix[1] and there exist algorithms which are guaranteed to find this solution [6, 7]. When the lower and upper bounds on the BLCP are finite, the BLCP can in fact be formulated as an equivalent LCP of twice the dimensionality of the original problem. A full derivation of this equivalence is shown in the appendix (supplementary materials).

There are many algorithms for solving (B)LCPs. Since our approach is not tied to a particular algorithm, we review some general properties of (B)LCP solvers. Optimized solvers can take advantage of sparsity in $z$. A zero entry in $z$ effectively cancels out a column in $M$. If $M$ is large, efficient solvers can avoid using $M$ directly, instead using a smaller $M'$ that is induced by the nonzero entries of $z$. The columns of $M'$ can be thought of as the "active" columns and the procedure of swapping columns in and out of $M'$ can be thought of as a pivoting operation, analogous to pivots in the simplex algorithm. Another important property of some (B)LCP algorithms is their ability to start from an initial guess at the solution (i.e., a "warm start"). If the initial guess is close to a solution, this can significantly reduce the solver's runtime.

Recently, Kim and Park [8] derived a connection between the BLCP and the Karush-Kuhn-Tucker (KKT) conditions for LARS. In particular, they noted the solution to the minimization problem in equation (1) has the form:

$$\underbrace{x}_{w} = \underbrace{(\Phi^T \Phi)^{-1} \Phi^T y}_{q} + \underbrace{(\Phi^T \Phi)^{-1}}_{M} \underbrace{(-c)}_{z},$$

where the vector $-c$ follows the constraints in equation (2) with $l_i = -\beta$ and $u_i = \beta$. Although we describe the equivalence between the BLCP and LARS optimality conditions using $M \equiv (\Phi^T \Phi)^{-1}$, the inverse can take place inside the BLCP algorithm and this operation is feasible and efficient as it is only done for the active columns of $\Phi$. Kim and Park [8] used a block pivoting algorithm, originally introduced by Júdice and Pires [6], for solving the Lasso. Their experiments show the block pivoting algorithm is significantly faster than both LARS and Feature Sign Search [9].

## 3  Previous Work

Recent work has emphasized feature selection as an important problem in reinforcement learning [10, 11]. Farahmand et al. [12] consider $L_2$ regularized RL. An $L_1$ regularized Bellman residual minimization algorithm was proposed by Loth et al. [13][2]. Johns and Mahadevan [14] investigate the combination of least squares temporal difference learning (LSTD) [2] with different variants of the matching pursuit algorithm [15, 16]. Petrik et al. [3] consider $L_1$ regularization in the context of approximate linear programming. Their approach offers some strong guarantees, but is not well-suited to noisy, sampled data.

The work most directly related to our own is that of Kolter and Ng [1]. They propose augmenting the LSTD algorithm with an $L_1$ regularization penalty. This results in the following $L_1$ *regularized linear fixed point* ($L_1$TD) problem:

$$w = \operatorname*{arg\,min}_{x \in \mathbb{R}^k} \frac{1}{2} \|\Phi x - (R + \gamma \Phi'^\pi w)\|_2^2 + \beta \|x\|_1. \tag{3}$$

Kolter and Ng derive a set of necessary and sufficient conditions characterizing the above fixed point[3] in terms of $\beta, w,$ and a vector $c$ of correlations between the features and the Bellman residual $T^\pi \hat{V} - \hat{V}$. More specifically, the correlation $c_i$ associated with feature $\varphi_i$ is given by:

$$c_i = \varphi_i^T (T^\pi \hat{V} - \hat{V}) = \varphi_i^T (R + \gamma \Phi'^\pi w - \Phi w). \tag{4}$$

Introducing the notation $\mathcal{I}$ to denote the set of indices of *active* features in the model (i.e., $\mathcal{I} = \{i : w_i \neq 0\}$), the fixed point optimality conditions can be summarized as follows:

$C_1$. All features in the active set share the same absolute correlation, $\beta$: $\forall i \in \mathcal{I}, |c_i| = \beta$.

$C_2$. Inactive features have less absolute correlation than active features: $\forall i \notin \mathcal{I}, |c_i| < \beta$.

$C_3$. Active features have correlations and weights agreeing in sign: $\forall i \in \mathcal{I}, \operatorname{sgn}(c_i) = \operatorname{sgn}(w_i)$.

Kolter and Ng show that it is possible to find the fixed point using an iterative procedure adapted from LARS. Their algorithm, LARS-TD, computes a sequence of fixed points, each of which satisfies the optimality conditions above for some intermediate $L_1$ parameter $\bar{\beta} \geq \beta$. Successive solutions decrease $\bar{\beta}$ and are computed in closed form by determining the point at which a feature must be added or removed in order to further decrease $\bar{\beta}$ without violating one of the fixed point requirements. The algorithm (as applied to action-value function approximation) is a special case of the algorithm presented in the appendix (see Fig. 2). Kolter and Ng prove that if $\Phi^T(\Phi - \gamma \Phi'^\pi)$ is a P-matrix, then for any $\beta \geq 0$, LARS-TD will find a solution to equation (3).

LARS-TD inherits many of the benefits and limitations of LARS. The fact that it traces an entire homotopy path can be quite helpful because it does not require committing to a particular value of $\beta$. On the other hand, the incremental nature of LARS may not be the most efficient solution for any single value of the regularization parameter, as shown by Lee et al. [9] and Kim and Park [8].

It is natural to employ LARS-TD in an iterative manner within the least squares policy iteration (LSPI) algorithm [17], as Kolter and Ng did. In this usage, however, many of the benefits of LARS are lost. When a new policy is selected in the policy iteration loop, LARS-TD must discard its solution from the previous policy and start an entirely new homotopy path, making the value of the homotopy path in this context not entirely clear. One might cross-validate a choice of regularization parameter by measuring the performance of the final policy, but this requires guessing a value of $\beta$ for all policies and then running LARS-TD up to this value for each policy. If a new value of $\beta$ is tried, all of the work done for the previous value must be discarded.

## 4   The $L_1$ Regularized Fixed Point as an LCP

We show that the optimality conditions for the $L_1$TD fixed point correspond to the solution of a (B)LCP. This reformulation allows for (1) new algorithms to compute the fixed point using (B)LCP solvers, and (2) a new guarantee on the uniqueness of a fixed point.

The $L_1$ regularized linear fixed point is described by a vector of correlations $c$ as defined in equation (4). We introduce the following variables:

$$A = \Phi^T (\Phi - \gamma \Phi'^\pi) \qquad\qquad b = \Phi^T R,$$

that allow equation (4) to be simplified as $c = b - Aw$. Assuming $A$ is a P-matrix, $A$ is invertible[4] [18] and we can write:

$$\underbrace{w}_{w} = \underbrace{A^{-1}b}_{q} + \underbrace{A^{-1}}_{M}\underbrace{(-c)}_{z}.$$

Consider a solution ($w$ and $z$) to the equation above where $z$ is bounded as in equation (2) with $l = -\boldsymbol{\beta}$ and $u = \boldsymbol{\beta}$ to specify a BLCP. It is easy to verify that coefficients $w$ satisfying this BLCP acheive the $L_1$TD optimality conditions as detailed in section 3. Thus, any appropriate solver for the BLCP($A^{-1}b, A^{-1}, -\boldsymbol{\beta}, \boldsymbol{\beta}$) can be thought of as a linear complementarity approach to solving for the $L_1$TD fixed point. We refer to this class of solvers as *LC-TD algorithms* and parameterize them as LC-TD($\Phi, \Phi'^{\pi}, R, \gamma, \beta$).

**Proposition 1** *If $A$ is a P-matrix, then for any $R$, the $L_1$ regularized linear fixed point exists, is unique, and will be found by a basic-set BLCP algorithm solving BLCP($A^{-1}b, A^{-1}, -\boldsymbol{\beta}, \boldsymbol{\beta}$).*

This proposition follows immediately from some basic BLCP results. We note that if $A$ is a P-matrix, so is $A^{-1}$ [18], that BLCPs for P-matrices have a unique solution for any $q$ ([7], Chp. 3), and that the the basic-set algorithm of Júdice and Pires [19] is guaranteed to find a solution to any BLCP with a P-matrix. This strengthens the theorem by Kolter and Ng [1], which guaranteed only that the LARS-TD algorithm would converge to *a* solution when $A$ is a P-matrix.

This connection to the LCP literature has practical benefits as well as theoretical ones. Decoupling the problem from the solver allows a variety of algorithms to be exploited. For example, the ability of many solvers to use a warm start during initialization offers a significant computational advantage over LARS-TD (which always begins with a null solution). In the experimental section of this paper, we demonstrate that the ability to use warm starts during policy iteration can significantly improve computational efficiency. We also find that (B)LCP solvers can be more robust than LARS-TD, an issue we address further in the appendix.

## 5 Modified Policy Iteration using LARS-TD and LC-TD

As mentioned in section 3, the advantages of LARS-TD as a homotopy method are less clear when it is used in a policy iteration loop since the homotopy path is traced only for specific policies. It is possible to incorporate greedy policy improvements into the LARS-TD loop, leading to a homotopy path for greedy policies. The greedy $L_1$ regularized fixed point equation is:

$$w = \arg\min_{x \in \mathbb{R}^k} \frac{1}{2}\|\Phi x - \max_{\pi}(R + \gamma\Phi'^{\pi}w)\|_2^2 + \beta\|x\|_1. \tag{5}$$

We propose a modification to LARS-TD called LARQ which, along with conditions $C_1$-$C_3$ in section 3, maintains an additional invariant:

    $C_4$. The current policy $\pi$ is greedy with respect to the current solution.

It turns out that we can change policies and avoid violating the LARS-TD invariants if we make policy changes at points where applying the Bellman operator yields the same value for both the old policy ($\pi$) and the new policy ($\pi'$): $T^\pi\hat{V} = T^{\pi'}\hat{V}$. The LARS-TD invariants all depend on the correlation of features with the residual $T^\pi\hat{V} - \hat{V}$ of the current solution. When the above equation is satisfied, the residual is equal for both policies. Thus, we can change policies at such points without violating any of the LARS-TD invariants. Due to space limitations, we defer a full presentation of the LARQ algorithm to the appendix.

When run to completion, LARQ provides a set of action-values that are the greedy fixed point for *all* settings of $\beta$. In principle, this is more flexible than LARS-TD with policy iteration because it produces these results in a single run of the algorithm. In practice, LARQ suffers two limitations.

The first is that it can be slow. LARS-TD enumerates every point at which the active set of features might change, a calculation that must be redone every time the active set changes. LARQ must do this as well, but it must also enumerate all points at which the greedy policy can change. For $k$ features and $n$ samples, LARS-TD must check $O(k)$ points, but LARQ must check $O(k+n)$ points. Even though LARS-TD will run multiple times within a policy iteration loop, the number of such iterations will typically be far fewer than the number of training data points. In practice, we have observed that LARQ runs several times slower than LARS-TD with policy iteration.

A second limitation of LARQ is that it can get "stuck." This occurs when the greedy policy for a particular $\beta$ is not well defined. In such cases, the algorithm attempts to switch to a new policy immediately following a policy change. This problem is not unique to LARQ. Looping is possible with most approximate policy iteration algorithms. What makes it particularly troublesome for LARQ is that there are few satisfying ways of addressing this issue without sacrificing the invariants.

To address these limitations, we present a compromise between LARQ and LARS-TD with policy iteration. The algorithm, LC-MPI, is presented as Algorithm 1. It avoids the cost of continually checking for policy changes by updating the policy only at a fixed set of values, $\beta^{(1)} \dots \beta^{(m)}$. Note that the $\beta$ values are in decreasing order with $\beta^{(1)}$ set to the maximum value (i.e., the point such that $w^{(1)}$ is the zero vector). At each $\beta^{(j)}$, the algorithm uses a policy iteration loop to (1) determine the current policy (greedy with respect to parameters $\hat{w}^{(j)}$), and (2) compute an approximate value function $\Phi w^{(j)}$ using LC-TD. The policy iteration loop terminates when $w^{(j)} \approx \hat{w}^{(j)}$ or some predefined number of iterations is exceeded. This use of LC-TD within a policy iteration loop will typically be quite fast because we can use the current feature set as a warm start. The warm start is indicated in Algorithm 1 by $supp(\hat{w}^{(j)})$, where the function $supp$ determines the support, or active elements, in $\hat{w}^{(j)}$; many (B)LCP solvers can use this information for initialization.

Once the policy iteration loop terminates for point $\beta^{(j)}$, LC-MPI simply begins at the next point $\beta^{(j+1)}$ by initializing the weights with the previous solution, $\hat{w}^{(j+1)} \leftarrow w^{(j)}$. This was found to be a very effective technique. As an alternative, we tested initializing $\hat{w}^{(j+1)}$ with the result of running LARS-TD with the greedy policy implicit in $w^{(j)}$ from the point $(\beta^{(j)}, w^{(j)})$ to $\beta^{(j+1)}$. This initialization method performed worse experimentally than the simple approach described above.

We can view LC-MPI as approximating LARQ's homotopy path since the two algorithms agree for any $\beta^{(j)}$ reachable by LARQ. However, LC-MPI is more efficient and avoids the problem of getting stuck. By compromising between the greedy updates of LARQ and the pure policy evaluation methods of LARS-TD and LC-TD, LC-MPI can be thought of as form of modified policy iteration [20]. The following table summarizes the properties of the algorithms described in this paper.

| | LARS-TD Policy Iteration | LC-TD Policy Iteration | LARQ | LC-MPI |
|---|---|---|---|---|
| Warm start for each new $\beta$ | N | N | Y | Y |
| Warm start for each new policy | N | Y | Y | Y |
| Greedy policy homotopy path | N | N | Y | Approximate |
| Robust to policy cycles | Y | Y | N | Y |

# 6 Experiments

We performed two types of experiments to highlight the potential benefits of (B)LCP algorithms. First, we used both LARS-TD and LC-TD within policy iteration. These experiments, which were run using a single value of the $L_1$ regularization parameter, show the benefit of warm starts for LC-TD. The second set of experiments demonstrates the benefit of using the LC-MPI algorithm. A single run of LC-MPI results in greedy policies for *multiple* values of $\beta$, allowing the use of cross-validation to pick the best policy. We show this is significantly more efficient than running policy iteration with either LARS-TD or LC-TD multiple times for different values of $\beta$. We discuss the details of the specific LCP solver we used in the appendix.

Both types of experiments were conducted on the 20-state chain [17] and mountain car [21] domains, the same problems tested by Kolter and Ng [1]. The chain MDP consists of two stochastic actions, left and right, a reward of one at each end of the chain, and $\gamma = 0.9$. One thousand samples were generated using 100 episodes, each consisting of 10 random steps. For features, we used 1000 Gaussian random noise features along with five equally spaced radial basis functions (RBFs) and a constant function. The goal in the mountain car MDP is to drive an underpowered car up a hill

---

**Algorithm 1** LC-MPI

---

**Inputs:**
  $\{s_i, a_i, r_i, s_i'\}_{i=1}^n$,   state transition and reward samples
  $\varphi : S \times A \to \mathbb{R}^k$,   state-action features
  $\gamma \in [0, 1)$,   discount factor
  $\{\beta^{(j)}\}_{j=1}^m$,   where $\beta^{(1)} = \max_l \left|\sum_{i=1}^n \varphi_l(s_i, a_i) r_i\right|, \beta^{(j)} < \beta^{(j-1)}$ for $j \in \{2, \dots, m\}$, and $\beta^{(m)} \geq 0$
  $\epsilon \in \mathbb{R}_+$  and  $T \in \mathbb{N}$,   termination conditions for policy iteration

**Initialization:**
  $\Phi \leftarrow [\varphi(s_1, a_1) \ \ \dots \ \ \varphi(s_n, a_n)]^T$,     $R \leftarrow [r_1 \ \ \dots \ \ r_n]^T$,     $w^{(1)} \leftarrow \mathbf{0}$

**for** $j = 2$ to $m$ **do**
    *// Initialize with the previous solution*
    $\hat{w}^{(j)} \leftarrow w^{(j-1)}$
    *// Policy iteration loop*
    **Loop:**
        *// Select greedy actions and form $\Phi'$*
        $\forall i : a_i' \leftarrow \arg\max_a \varphi(s_i', a)^T \hat{w}^{(i)}$
        $\Phi' \leftarrow [\varphi(s_1', a_1') \ \ \dots \ \ \varphi(s_n', a_n')]^T$
        *// Solve the LC-TD problem using a (B)LCP solver with a warm start*
        $w^{(j)} \leftarrow \text{LC-TD}(\Phi, \Phi', R, \gamma, \beta^{(j)})$ with warm start $supp(\hat{w}^{(j)})$
        *// Check for termination*
        **if**  $(\|w^{(j)} - \hat{w}^{(j)}\|_2 \leq \epsilon)$  or  (# iterations $\geq T$)
            **then**  break loop
            **else**  $\hat{w}^{(j)} \leftarrow w^{(j)}$
**Return**  $\{w^{(j)}\}_{j=1}^m$

---

by building up momentum. The domain is continuous, two dimensional, and has three actions. We used $\gamma = 0.99$ and 155 radial basis functions (apportioned as a two dimensional grid of $1, 2, 3, 4, 5,$ $6,$ and 8 RBFs) and one constant function for features. Samples were generated using 75 episodes where each episode started in a random start state, took random actions, and lasted at most 20 steps.

## 6.1   Policy Iteration

To compare LARS-TD and LC-TD when employed within policy iteration, we recorded the *number of steps* used during each round of policy iteration, where a *step* corresponds to a change in the active feature set. The computational complexity per step of each algorithm is similar; therefore, we used the average number of steps per policy as a metric for comparing the algorithms. Policy iteration was run either until the solution converged or 15 rounds were exceeded. This process was repeated 10 times for 11 different values of $\beta$. We present the results from these experiments in the first two columns of Table 1. The two algorithms performed similarly for the chain MDP, but LC-TD used significantly fewer steps for the mountain car MDP. Figure 1 shows plots for the number of steps used for each round of policy iteration for a single (typical) trial. Notice the declining trend for LC-TD; this is due to the warm starts requiring fewer steps to find a solution. The plot for the chain MDP shows that LC-TD uses many more steps in the first round of policy iteration than does LARS-TD. Lastly, in the trials shown in Figure 1, policy iteration using LC-TD converged in six iterations whereas it did not converge at all when using LARS-TD. This was due to LARS-TD producing solutions that violate the $L_1$ TD optimality conditions. We discuss this in detail in appendix A.5.

## 6.2   LC-MPI

When LARS-TD and LC-TD are used as subroutines within policy iteration, the process ends at a single value of the $L_1$ regularization parameter $\beta$. The policy iteration loop must be rerun to consider different values of $\beta$. In this section, we show how much computation can be saved by running LC-MPI once (to produce $m$ greedy policies, each at a different value of $\beta$) versus running policy iteration $m$ separate times. The third column in Table 1 shows the average number of algorithm steps per policy for LC-MPI. As expected, there is a significant reduction in complexity by using LC-MPI for both domains. In the appendix, we give a more detailed example of how cross-validation can be

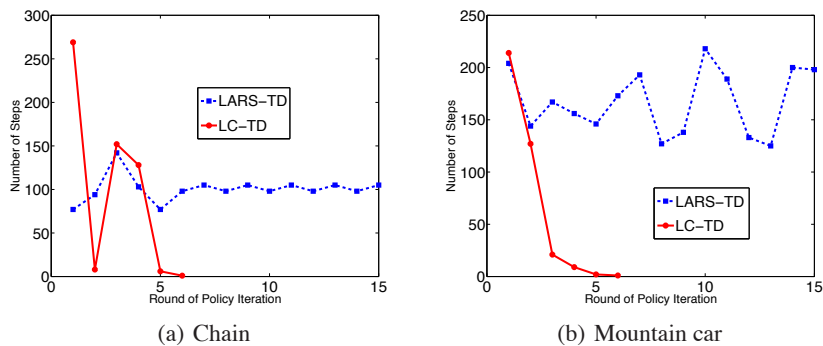

| (a) Chain | (b) Mountain car |

Figure 1: Number of steps used by algorithms LARS-TD and LC-TD during each round of policy iteration for a typical trial. For LC-TD, note the decrease in steps due to warm starts.

| Domain | LARS-TD, PI | LC-TD, PI | LC-MPI |
|---|---|---|---|
| Chain | $73 \pm 13$ | $77 \pm 11$ | $24 \pm 11$ |
| Mountain car | $214 \pm 33$ | $116 \pm 22$ | $21 \pm 5$ |

Table 1: Average number of algorithm steps per policy.

used to select a good value of the regularization parameter. We also offer some additional comments on the robustness of the LARS-TD algorithm.

## 7  Conclusions

In this paper, we proposed formulating the $L_1$ regularized linear fixed point problem as a linear complementarity problem. We showed the LCP formulation leads to a stronger theoretical guarantee in terms of the solution's uniqueness than was previously shown. Furthermore, we demonstrated that the "warm start" ability of LCP solvers can accelerate the computation of the $L_1$TD fixed point when initialized with the support set of a related problem. This was found to be particularly effective for policy iteration problems when the set of active features does not change significantly from one policy to the next.

We proposed the LARQ algorithm as an alternative to LARS-TD. The difference between these algorithms is that LARQ incorporates greedy policy improvements inside the homotopy path. The advantage of this "greedy" homotopy path is that it provides a set of action-values that are a greedy fixed point for all settings of the $L_1$ regularization parameter. However, this additional flexibility comes with increased computational complexity. As a compromise between LARS-TD and LARQ, we proposed the LC-MPI algorithm which only maintains the LARQ invariants at a fixed set of values. The key to making LC-MPI efficient is the use of warm starts by using an LCP algorithm.

There are several directions for future work. An interesting question is whether there is a natural way to incorporate policy improvement directly within the LCP formulation. Another concern for $L_1$TD algorithms is a better characterization of the conditions under which solutions exist and can be found efficiently. In previous work, Kolter and Ng [1] indicated the P-matrix property can always hold provided enough $L_2$ regularization is added to the problem. While this is possible, it also decreases the sparsity of the solution; therefore, it would be useful to find other techniques for guaranteeing convergence while maintaining sparsity.

### Acknowledgments

This work was supported by the National Science Foundation (NSF) under Grant #0937060 to the Computing Research Association for the CIFellows Project, NSF Grant IIS-0713435, and DARPA CSSG HR0011-06-1-0027. Any opinions, findings, and conclusions or recommendations expressed in this material are those of the authors and do not necessarily reflect the views of the National Science Foundation or the Computing Research Association.

## Footnotes

[1]A P-matrix is a matrix for which all principal minors are positive.

[2]Loth et al. claim to adapt LSTD to $L_1$ regularization, but in fact describe a Bellman residual minimization algorithm and not a fixed point calculation.

[3] For fixed $w$, the RHS of equation (3) is a convex optimization problem; a sufficient condition for optimality of some vector $x^*$ is that the zero vector is in the subdifferential of the RHS at $x^*$. The fixed point conditions follow from the equality between the LHS and RHS.

[4]Even when $A$ is not invertible, we can still use a BLCP solver as long as the principal submatrix of $A$ associated with the active features is invertible. As with LARS-TD, the inverse only occurs for this principal submatrix. In fact, we discuss in the appendix how one need never explicitly compute $A$. Alternatively, we can convert the BLCP to an LCP (appendix A.1) thereby avoiding $A^{-1}$ in the parameterization of the problem.

# References

[1] J. Kolter and A. Ng. Regularization and feature selection in least-squares temporal difference learning. In *Proc. ICML*, pages 521–528, 2009.

[2] S. Bradtke and A. Barto. Linear least-squares algorithms for temporal difference learning. *Machine Learning*, 22(1-3):33–57, 1996.

[3] M. Petrik, G. Taylor, R. Parr, and S. Zilberstein. Feature selection using regularization in approximate linear programs for Markov decision processes. In *To appear in Proc. ICML*, 2010.

[4] R. Tibshirani. Regression shrinkage and selection via the Lasso. *Journal of the Royal Statistical Society. Series B (Methodological)*, 58(1):267–288, 1996.

[5] B. Efron, T. Hastie, I. Johnstone, and R. Tibshirani. Least angle regression. *The Annals of Statistics*, 32(2):407–451, 2004.

[6] J. Júdice and F. Pires. A block principal pivoting algorithm for large-scale strictly monotone linear complementarity problems. *Computers and Operations Research*, 21(5):587–596, 1994.

[7] K. Murty. *Linear Complementarity, Linear and Nonlinear Programming*. Heldermann Verlag, 1988.

[8] J. Kim and H. Park. Fast active-set-type algorithms for $L_1$-regularized linear regression. In *Proc. AISTAT*, pages 397–404, 2010.

[9] H. Lee, A. Battle, R. Raina, and A. Ng. Efficient sparse coding algorithms. In *Advances in Neural Information Processing Systems 19*, pages 801–808, 2007.

[10] S. Mahadevan and M. Maggioni. Proto-value functions: A Laplacian framework for learning representation and control in Markov decision processes. *JMLR*, 8:2169–2231, 2007.

[11] R. Parr, L. Li, G. Taylor, C. Painter-Wakefield, and M. Littman. An analysis of linear models, linear value-function approximation, and feature selection for reinforcement learning. In *Proc. ICML*, 2008.

[12] A. Farahmand, M. Ghavamzadeh, C. Szepesvári, and S. Mannor. Regularized fitted Q-iteration for planning in continuous-space Markovian decision problems. In *Proc. ACC*. IEEE Press, 2009.

[13] M. Loth, M. Davy, and P. Preux. Sparse temporal difference learning using LASSO. In *IEEE International Symposium on Approximate Dynamic Programming and Reinforcement Learning*, 2007.

[14] J. Johns and S. Mahadevan. Sparse approximate policy evaluation using graph-based basis functions. Technical Report UM-CS-2009-041, University of Massachusetts Amherst, Department of Computer Science, 2009.

[15] S. Mallat and Z. Zhang. Matching pursuits with time-frequency dictionaries. *IEEE Transactions on Signal Processing*, 41(12):3397–3415, 1993.

[16] Y. Pati, R. Rezaiifar, and P. Krishnaprasad. Orthogonal matching pursuit: Recursive function approximation with applications to wavelet decomposition. In *Proceedings of the 27th Annual Asilomar Conference on Signals, Systems, and Computers*, volume 1, pages 40–44, 1993.

[17] M. Lagoudakis and R. Parr. Least-squares policy iteration. *Journal of Machine Learning Research*, 4:1107–1149, 2003.

[18] S. Lee and H. Seol. A survey on the matrix completion problem. *Trends in Mathematics*, 4(1):38–43, 2001.

[19] J. Júdice and F. Pires. Basic-set algorithm for a generalized linear complementarity problem. *Journal of Optimization Theory and Applications*, 74(3):391–411, 1992.

[20] M. Puterman and M. Shin. Modified policy iteration algorithms for discounted Markov decision problems. *Management Science*, 24(11), 1978.

[21] R. Sutton and A. Barto. *Reinforcement Learning: An Introduction*. MIT Press, 1998.

